# Exponential Family Harmoniums
# with an Application to Information Retrieval

**Max Welling & Michal Rosen-Zvi**
Information and Computer Science
University of California
Irvine CA 92697-3425 USA
*welling@ics.uci.edu*

**Geoffrey Hinton**
Department of Computer Science
University of Toronto
Toronto, 290G M5S 3G4, Canada
*hinton@cs.toronto.edu*

## Abstract

Directed graphical models with one layer of observed random variables and one or more layers of hidden random variables have been the dominant modelling paradigm in many research fields. Although this approach has met with considerable success, the causal semantics of these models can make it difficult to infer the posterior distribution over the hidden variables. In this paper we propose an alternative two-layer model based on exponential family distributions and the semantics of undirected models. Inference in these "exponential family harmoniums" is fast while learning is performed by minimizing contrastive divergence. A member of this family is then studied as an alternative probabilistic model for latent semantic indexing. In experiments it is shown that they perform well on document retrieval tasks and provide an elegant solution to searching with keywords.

## 1  Introduction

Graphical models have become the basic framework for generative approaches to probabilistic modelling. In particular models with latent variables have proven to be a powerful way to capture hidden structure in the data. In this paper we study the important subclass of models with one layer of observed units and one layer of hidden units.

Two-layer models can be subdivided into various categories depending on a number of characteristics. An important property in that respect is given by the semantics of the graphical model: either directed (Bayes net) or undirected (Markov random field). Most two-layer models fall in the first category or are approximations derived from it: mixtures of Gaussians (MoG), probabilistic PCA (pPCA), factor analysis (FA), independent components analysis (ICA), sigmoid belief networks (SBN), latent trait models, latent Dirichlet allocation (LDA, otherwise known as multinomial PCA, or mPCA) [1], exponential family PCA (ePCA), probabilistic latent semantic indexing (pLSI) [6], non-negative matrix factorization (NMF), and more recently the multiple multiplicative factor model (MMF) [8].

Directed models enjoy important advantages such as easy (ancestral) sampling and easy handling of unobserved attributes under certain conditions. Moreover, the semantics of

directed models dictates marginal independence of the latent variables, which is a suitable modelling assumption for many datasets. However, it should also be noted that directed models come with an important disadvantage: inference of the posterior distribution of the latent variables given the observations (which is, for instance, needed within the context of the EM algorithm) is typically intractable resulting in approximate or slow iterative procedures. For important applications, such as latent semantic indexing (LSI), this drawback may have serious consequences since we would like to swiftly search for documents that are similar in the latent topic space.

A type of two-layer model that has not enjoyed much attention is the undirected analogue of the above described family of models. It was first introduced in [10] where it was named "harmonium". Later papers have studied the harmonium under various names (the "combination machine" in [4] and the "restricted Boltzmann machine" in [5]) and turned it into a practical method by introducing efficient learning algorithms. Harmoniums have only been considered in the context of discrete binary variables (in both hidden and observed layers), and more recently in the Gaussian case [7]. The first contribution of this paper is to extend harmoniums into the exponential family which will make them much more widely applicable.

Harmoniums also enjoy a number of important advantages which are rather orthogonal to the properties of directed models. Firstly, their product structure has the ability to produce distributions with very sharp boundaries. Unlike mixture models, adding a new expert may decrease or increase the variance of the distribution, which may be a major advantage in high dimensions. Secondly, unlike directed models, inference in these models is very fast, due to the fact that the latent variables are conditionally independent given the observations. Thirdly, the latent variables of harmoniums produce distributed representations of the input. This is much more efficient than the "grandmother-cell" representation associated with mixture models where each observation is generated by a single latent variable. Their most important disadvantage is the presence of a global normalization factor which complicates both the evaluation of probabilities of input vectors[1] and learning free parameters from examples. The second objective of this paper is to show that the introduction of contrastive divergence has greatly improved the efficiency of learning and paved the way for large scale applications.

Whether a directed two-layer model or a harmonium is more appropriate for a particular application is an interesting question that will depend on many factors such as prior (conditional) independence assumptions and/or computational issues such as efficiency of inference. To expose the fact that harmoniums can be viable alternatives to directed models we introduce an entirely new probabilistic extension of latent semantic analysis (LSI) [3] and show its usefulness in various applications. We do not want to claim superiority of harmoniums over their directed cousins, but rather that harmoniums enjoy rather different advantages that deserve more attention and that may one day be combined with the advantages of directed models.

## 2   Extending Harmoniums into the Exponential Family

Let $x_i$, $i = 1...M_x$ be the set of observed random variables and $h_j$, $j = 1...M_h$ be the set of hidden (latent) variables. Both $x$ and $h$ can take values in either the continuous or the discrete domain. In the latter case, each variable has states $a = 1...D$.

To construct an exponential family harmonium (EFH) we first choose $M_x$ independent distributions $p_i(x_i)$ for the observed variables and $M_h$ independent distributions $p_j(h_j)$

for the hidden variables from the exponential family and combine them multiplicatively,

$$p(\{x_i\}) = \prod_{i=1}^{M_x} r_i(x_i) \exp\Big[\sum_a \theta_{ia} f_{ia}(x_i) - A_i(\{\theta_{ia}\})\Big] \tag{1}$$

$$p(\{h_j\}) = \prod_{j=1}^{M_h} s_j(h_j) \exp\Big[\sum_b \lambda_{jb} g_{jb}(h_j) - B_j(\{\lambda_{jb}\})\Big] \tag{2}$$

where $\{f_{ia}(x_i), g_{jb}(h_j)\}$ are the sufficient statistics for the models (otherwise known as features), $\{\theta_{ia}, \lambda_{jb}\}$ the canonical parameters of the models and $\{A_i, B_j\}$ the log-partition functions (or log-normalization factors). In the following we will consider $\log(r_i(x_i))$ and $\log(s_j(h_j))$ as additional features multiplied by a constant.

Next, we couple the random variables in the log-domain by the introduction of a quadratic interaction term,

$$p(\{x_i, h_j\}) \propto \exp\Big[\sum_{ia} \theta_{ia} f_{ia}(x_i) + \sum_{jb} \lambda_{jb} g_{jb}(h_j) + \sum_{ijab} W_{ia}^{jb} f_{ia}(x_i) g_{jb}(h_j)\Big] \tag{3}$$

Note that we did not write the log-partition function for this joint model in order to indicate our inability to compute it in general. For some combinations of exponential family distributions it may be necessary to restrict the domain of $W_{ia}^{jb}$ in order to maintain normalizability of the joint probability distribution (e.g. $W_{ia}^{jb} \leq 0$ or $W_{ia}^{jb} \geq 0$). Although we could also have mutually coupled the observed variables (and/or the hidden variables) using similar interaction terms we refrain from doing so in order to keep the learning and inference procedures efficient. Consequently, by this construction the conditional probability distributions are a product of *independent* distributions in the exponential family with shifted parameters,

$$p(\{x_i\}|\{h_j\}) = \prod_{i=1}^{M_x} \exp\Big[\sum_a \hat{\theta}_{ia} f_{ia}(x_i) - A_i(\{\hat{\theta}_{ia}\})\Big] \quad \hat{\theta}_{ia} = \theta_{ia} + \sum_{jb} W_{ia}^{jb} g_{jb}(h_j) \tag{4}$$

$$p(\{h_j\}|\{x_i\}) = \prod_{j=1}^{M_h} \exp\Big[\sum_b \hat{\lambda}_{jb} g_{jb}(h_j) - B_j(\{\hat{\lambda}_{jb}\})\Big] \quad \hat{\lambda}_{jb} = \lambda_{jb} + \sum_{ia} W_{ia}^{jb} f_{ia}(x_i) \tag{5}$$

Finally, using the following identity, $\sum_y \exp \sum_a \theta_a f_a(y) = \exp A(\{\theta_a\})$ we can also compute the marginal distributions of the observed and latent variables,

$$p(\{x_i\}) \propto \exp\Big[\sum_{ia} \theta_{ia} f_{ia}(x_i) + \sum_j B_j(\{\lambda_{jb} + \sum_{ia} W_{ia}^{jb} f_{ia}(x_i)\})\Big] \tag{6}$$

$$p(\{h_j\}) \propto \exp\Big[\sum_{jb} \lambda_{jb} g_{jb}(h_j) + \sum_i A_i(\{\theta_{ia} + \sum_{jb} W_{ia}^{jb} g_{jb}(h_j)\})\Big] \tag{7}$$

Note that 1) we can only compute the marginal distributions up to the normalization constant and 2) in accordance with the semantics of undirected models, there is no marginal independence between the variables (but rather conditional independence).

## 2.1 Training EF-Harmoniums using Contrastive Divergence

Let $\tilde{p}(\{x_i\})$ denote the data distribution (or the empirical distribution in case we observe a finite dataset), and $p$ the model distribution. Under the maximum likelihood objective the learning rules for the EFH are conceptually simple[2],

$$\delta\theta_{ia} \propto \langle f_{ia}(x_i)\rangle_{\tilde{p}} - \langle f_{ia}(x_i)\rangle_p \qquad \delta\lambda_{jb} \propto \langle B'_{jb}(\hat{\lambda}_{jb})\rangle_{\tilde{p}} - \langle B'_{jb}(\hat{\lambda}_{jb})\rangle_p \tag{8}$$

$$\delta W_{ij}^{ab} \propto \langle f_{ia}(x_i) B'_{jb}(\hat{\lambda}_{jb}) \rangle_{\tilde{p}} - \langle f_{ia}(x_i) B'_{jb}(\hat{\lambda}_{jb}) \rangle_p \tag{9}$$

where we have defined $B'_{jb} = \partial B_j(\hat{\lambda}_{jb})/\partial \hat{\lambda}_{jb}$ with $\hat{\lambda}_{jb}$ defined in Eqn.5. One should note that these learning rules are changing the parameters in an attempt to match the expected sufficient statistics of the data distribution and the model distribution (while maximizing entropy). Their simplicity is somewhat deceptive, however, since the averages $\langle \cdot \rangle_p$ are intractable to compute analytically and Markov chain sampling or mean field calculations are typically wheeled out to approximate them. Both have difficulties: mean field can only represent one mode of the distribution and MCMC schemes are slow and suffer from high variance in their estimates.

In the case of binary harmoniums (restricted BMs) it was shown in [5] that contrastive divergence has the potential to greatly improve on the efficiency and reduce the variance of the estimates needed in the learning rules. The idea is that instead of running the Gibbs sampler to its equilibrium distribution we initialize Gibbs samplers on each data-vector and run them for only one (or a few) steps in parallel. Averages $\langle \cdot \rangle_p$ in the learning rules Eqns.8,9 are now replaced by averages $\langle \cdot \rangle_{p_{\mathbf{CD}}}$ where $p_{\mathbf{CD}}$ is the distribution of samples that resulted from the truncated Gibbs chains. This idea is readily generalized to EFHs. Due to space limitations we refer to [5] for more details on contrastive divergence learning[3]. Deterministic learning rules can also be derived straightforwardly by generalizing the results described in [12] to the exponential family.

## 3 A Harmonium Model for Latent Semantic Indexing

To illustrate the new possibilities that have opened up by extending harmoniums to the exponential family we will next describe a novel model for latent semantic indexing (LSI). This will represent the undirected counterpart of pLSI [6] and LDA [1].

One of the major drawbacks of LSI is that inherently discrete data (word counts) are being modelled with variables in the continuous domain. The power of LSI on the other hand is that it provides an efficient mapping of the input data into a lower dimensional (continuous) latent space that has the effect of de-noising the input data and inferring semantic relationships among words. To stay faithful to this idea and to construct a probabilistic model on the correct (discrete) domain we propose the following EFH with continuous latent topic variables, $h_j$, and discrete word-count variables, $x_{ia}$,

$$p(\{h_j\}|\{x_{ia}\}) = \prod_{j=1}^{M_h} \mathcal{N}_{h_j}[\sum_{ia} W_{ia}^j x_{ia}, 1] \tag{10}$$

$$p(\{x_{ia}\}|\{h_j\}) = \prod_{i=1}^{M_x} \mathcal{S}_{\{x_{ia}\}}[\alpha_{ia} + \sum_j h_j W_{ia}^j] \tag{11}$$

Note that $\{x_{ia}\}$ represent indicator variables satisfying $\sum_a x_{ia} = 1 \ \forall i$, where $x_{ia} = 1$ means that word "$i$" in the vocabulary was observed "$a$" times. $\mathcal{N}_h[\mu, \sigma]$ denotes a normal distribution with mean $\mu$ and *std.* $\sigma$ and $\mathcal{S}_{\{x_a\}}[\gamma_a] \propto \exp(\sum_{a=1}^D \gamma_a x_a)$ is the softmax function defining a probability distribution over $x$. Using Eqn.6 we can easily deduce the marginal distribution of the input variables,

$$p(\{x_{ia}\}) \propto \exp[\sum_{ia} \alpha_{ia} x_{ia} + \frac{1}{2} \sum_j (\sum_{ia} W_{ia}^j x_{ia})^2] \tag{12}$$

We observe that the role of the components $W_{ia}^j$ is that of templates or prototypes: input vectors $x_{ia}$ with large inner products $\sum_{ia} W_{ia}^j x_{ia} \; \forall j$ will have high probability under this model. Just like pLSI and LDA can be considered as natural generalizations of factor analysis (which underlies LSI) into the class of directed models on the discrete domain, the above model can be considered as the natural generalization of factor analysis into class of *undirected* models on the discrete domain. This idea is supported by the result that the same model with Gaussian units in both hidden and observed layers is in fact equivalent to factor analysis [7].

### 3.1 Identifiability

From the form of the marginal distribution Eqn.12 we can derive a number of transformations of the parameters that will leave the distribution invariant. First we note that the components $W_{ia}^j$ can be rotated and mirrored arbitrarily in latent space[4]: $W_{ia}^j \to \sum_k U^{jk} W_{ia}^k$ with $U^T U = I$. Secondly, we note that observed variables $x_{ia}$ satisfy a constraint, $\sum_a x_{ia} = 1 \; \forall i$. This results in a combined shift invariance for the components $W_{ia}^j$ and the offsets $\alpha_{ia}$. Taken together, this results in the following set of transformations,

$$W_{ia}^j \to \sum_k U^{jk}(W_{ia}^k + V_i^k) \qquad \alpha_{ia} \to (\alpha_{ia} + \beta_i) - \sum_j (\sum_l V_l^j)(W_{ia}^j) \qquad (13)$$

where $U^T U = I$. Although these transformations leave the marginal distribution over the observable variables invariant, they do change the latent representation and as such may have an impact on retrieval performance (if we use a fixed similarity measure between topic representations of documents). To fix the spurious degrees of freedom we have chosen to impose conditions on the representations in latent space: $h_j^n = \sum_{ia} W_{ia}^j x_{ia}^n$. First, we center the latent representations which has the effect of minimizing the "activity" of the latent variables and moving as much log-probability as possible to the constant component $\alpha_{ia}$. Next we align the axes in latent space with the eigen-directions of the latent covariance matrix. This has the effect of approximately *decorrelating* the marginal latent activities. This follows because the marginal distribution in latent space can be approximated by: $p(\{h_j\}) \approx \sum_n \prod_j \mathcal{N}_{h_j}[\sum_{ia} W_{ia}^j x_{ia}^n, 1]/N$ where we have used Eqn.10 and replaced $p(\{x_{ia}\})$ by its empirical distribution. Denoting by $\mu$ and $\Sigma = U^T \Lambda U$ the sample mean and sample covariance of $\{h_j^n\}$, it is not hard to show that the following transformation will have the desired effect[5]:

$$W_{ia}^j \to \sum_k U^{jk}\left(W_{ia}^k - \frac{1}{M_x}\mu^k\right) \qquad \alpha_{ia} \to \alpha_{ia} + \sum_j \mu^j W_{ia}^j \qquad (14)$$

One could go one step further than the de-correlation process described above by introducing covariances $\Sigma$ in the conditional Gaussian distribution of the latent variables Eqn.10. This would not result in a more general model because the effect of this on the marginal distribution over the observed variables is given by: $W_{ia}^j \to \sum_k K^{jk} W_{ia}^k \quad KK^T = \Sigma$. However, the extra freedom can be used to define axes in latent space for which the projected data become approximately *independent* and have the same scale in all directions.

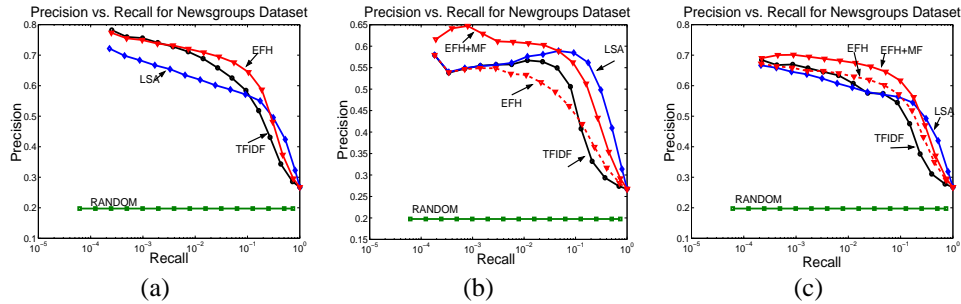

Figure 1: Precision-recall curves when the query was (a) entire documents, (b) 1 keyword, (c) 2 keywords for the EFH with and without 10 MF iterations, LSI , TFIDF weighted words and random guessing. PR curves with more keywords looked very similar to (c). A marker at position $k$ (counted from the left along a curve) indicates that $2^{k-1}$ documents were retrieved.

## 4    Experiments

**Newsgroups:**
We have used the reduced version of the "20newsgroups" dataset prepared for MATLAB by Roweis[6]. Documents are presented as 100 dimensional binary occurrence vectors and tagged as a member of 1 out of 4 domains. Documents contain approximately 4% of the words, averaged across the 16242 postings.

An EFH model with 10 latent variables was trained on 12000 training cases using stochastic gradient descent on mini-batches of 1000 randomly chosen documents (training time approximately 1 hour on a 2GHz PC). A momentum term was added to speed up convergence. To test the quality of the trained model we mapped the remaining 4242 query documents into latent space using $h_j = \sum_{ia} W_{ia}^j x_{ia}$ and where $\{W_{ia}^j, \alpha_{ia}\}$ were "gauged" as in Eqns.14. Precision-recall curves were computed by comparing training and query documents using the usual "cosine coefficient" (cosine of the angle between documents) and reporting success when the retrieved document was in the same domain as the query (results averaged over all queries). In figure 1a we compare the results with LSI (also 10 dimensions) [3] where we preprocessed the data in the standard way ($x \rightarrow log(1 + x)$ and entropy weighting of the words) and to similarity in word space using TF-IDF weighting of the words. In figure 1b,c we show PR curves when only 1 or 2 keywords were provided corresponding to randomly observed words in the query document. The EFH model allows a principled way to deal with unobserved entries by inferring them using the model (in all other methods we insert 0 for the unobserved entries which corresponds to ignoring them). We have used a few iterations of mean field to achieve that: $\hat{x}_{ia} \rightarrow \exp\left[\sum_{jb}(\sum_k W_{ia}^k W_{jb}^k + \alpha_{jb})\hat{x}_{jb}\right]/\gamma_i$ where $\gamma_i$ is a normalization constant and where $\hat{x}_{ia}$ represent probabilities: $\hat{x}_{ia} \in [0, 1]$, $\sum_{a=1}^D \hat{x}_{ia} = 1 \, \forall i$. We note that this is still highly efficient and achieves a significant improvement in performance. In all cases we find that without any preprocessing or weighting EFH still outperforms the other methods except when large numbers of documents were retrieved.

In the next experiment we compared performance of EFH, LSI and LDA by training models on a random subset of 15430 documents with 5 and 10 latent dimensions (this was found to be close to optimal for LDA). The EFH and LSI models were trained as in the previous experiment while the training and testing details[7] for LDA can be found in [9]. For the remaining test documents we clamped a varying number of observed words and

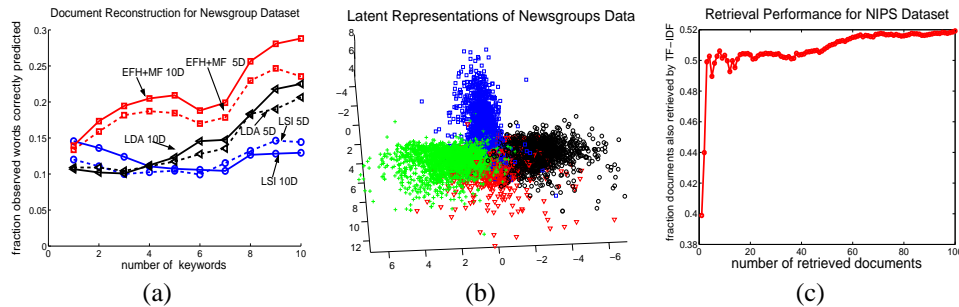

Figure 2: (a) Fraction of observed words that was correctly observed by EFH, LSI and LDA using 5 and 10 latent variables when we vary the number of keywords (observed words that were "clamped"), (b) latent 3-D representations of newsgroups data, (c) Fraction of documents retrieved by EFH on the NIPS dataset which was also retrieved by the TF-IDF method.

asked the models to predict the remaining observed words in the documents by computing the probabilities for all words in the vocabulary to be present and ranking them (see previous paragraph for details). By comparing the list of the $R$ remaining observed words in the document with the top-R ranked inferred words we computed the fraction of correctly predicted words. The results are shown in figure 2a as a function of the number of clamped words. To provide anecdotal evidence that EFH can infer semantic relationships we clamped the words 'drive' 'driver' and 'car' which resulted in: 'car' 'drive' 'engine' 'dealer' 'honda' 'bmw' 'driver' 'oil' as the most probable words in the documents. Also, clamping 'pc' 'driver' and 'program' resulted in: 'windows' 'card' 'dos' 'graphics' 'software' 'pc' 'program' 'files'.

**NIPS Conference Papers:**
Next we trained a model with 5 latent dimensions on the NIPS dataset[8] which has a large vocabulary size (13649 words) and contains 1740 documents of which 1557 were used for training and 183 for testing. Count values were redistributed in 12 bins. The array $W$ contains therefore $5 \cdot 13649 \cdot 12 = 818940$ parameters. Training was completed in the order of a few days. Due to the lack of document labels it is hard to assess the quality of the trained model. We choose to compare performance on document retrieval with the "golden standard": cosine similarity in TF-IDF weighted word space. In figure 2c we depict the fraction of documents retrieved by EFH that was also retrieved by TF-IDF as we vary the number of retrieved documents. This correlation is indeed very high but note that EFH computes similarity in a 5-D space while TF-IDF computes similarity in a 13649-D space.

## 5  Discussion

The main point of this paper was to show that there is a flexible family of 2-layer probabilistic models that represents a viable alternative to 2-layer causal (directed) models. These models enjoy very different properties and can be trained efficiently using contrastive divergence. As an example we have studied an EFH alternative for latent semantic indexing where we have found that the EFH has a number of favorable properties: fast inference allowing fast document retrieval and a principled approach to retrieval with keywords. These were preliminary investigations and it is likely that domain specific adjustments such as a more intelligent choice of features or parameterization could further improve performance.

Previous examples of EFH include the original harmonium [10], Gaussian variants thereof [7], and the PoT model [13] which couples a gamma distribution with the covariance of a

normal distribution. Some exponential family extensions of general Boltzmann machines were proposed in [2], [14], but they do not have the bipartite structure that we study here. While the components of the Gaussian-multinomial EFH act as prototypes or templates for highly probable input vectors, the components of the PoT act as *constraints* (i.e. input vectors with large inner product have low probability). This can be traced back to the shape of the non-linearity $B$ in Eqn.6. Although by construction $B$ must be convex (it is the log-partition function), for large input values it can both be positive (prototypes, e.g. $B(x) = x^2$) or negative (constraints, e.g. $B(x) = -log(1 + x)$). It has proven difficult to jointly model both prototypes and constraints in the this formalism except for the fully Gaussian case [11]. A future challenge is therefore to start the modelling process with the desired non-linearity and to subsequently introduce auxiliary variables to facilitate inference and learning.

## Footnotes

[1]However, it is easy to compute these probabilities up to a constant so it is possible to *compare* probabilities of data-points.

[2]These learning rules are derived by taking derivatives of the log-likelihood objective using Eqn.6.

[3]Non-believers in contrastive divergence are invited to simply run the the Gibbs sampler to equilibrium before they do an update of the parameters. They will find that due to the special bipartite structure of EFHs learning is still more efficient than for general Boltzmann machines.

[4]Technically we call this the Euclidean group of transformations.

[5]Some spurious degrees of freedom remain since shifts $\beta_i$ and shifts $V_i^j$ that satisfy $\sum_i V_i^j = 0$ will not affect the projection into latent space. One could decide to fix the remaining degrees of freedom by for example requiring that components are as small as possible in $L_2$ norm (subject to the constraint $\sum_i V_i^j = 0$), leading to the further shifts, $W_{ia}^j \to W_{ia}^j - \frac{1}{D}\sum_a W_{ia}^j + \frac{1}{DM_x}\sum_{ia} W_{ia}^j$ and $\alpha_{ia} \to \alpha_{ia} - \frac{1}{D}\sum_a \alpha_{ia}$.

[6]http://www.cs.toronto.edu/∼roweis/data.html

[7]The approximate inference procedure was implemented using Gibbs sampling.

[8]Obtained from http://www.cs.toronto.edu/~roweis/data.html.

## References

[1] D. M. Blei, A. Y. Ng, and M. I. Jordan. Latent Dirichlet allocation. *Journal of Machine Learning Research*, 3:993–1022, 2003.

[2] C.K.I.Williams. Continuous valued Boltzmann machines. Technical report, 1993.

[3] S.C. Deerwester, S.T. Dumais, T.K. Landauer, G.W. Furnas, and R.A. Harshman. Indexing by latent semantic analysis. *Journal of the American Society of Information Science*, 41(6):391–407, 1990.

[4] Y. Freund and D. Haussler. Unsupervised learning of distributions of binary vectors using 2-layer networks. In *Advances in Neural Information Processing Systems*, volume 4, pages 912–919, 1992.

[5] G.E. Hinton. Training products of experts by minimizing contrastive divergence. *Neural Computation*, 14:1771–1800, 2002.

[6] Thomas Hofmann. Probabilistic latent semantic analysis. In *Proc. of Uncertainty in Artificial Intelligence, UAI'99*, Stockholm, 1999.

[7] T. K. Marks and J. R. Movellan. Diffusion networks, products of experts, and factor analysis. Technical Report UCSD MPLab TR 2001.02, University of California San Diego, 2001.

[8] B. Marlin and R. Zemel. The multiple multiplicative factor model for collaborative filtering. In *Proceedings of the 21st International Conference on Machine Learning*, volume 21, 2004.

[9] M. Rosen-Zvi, T. Griffiths, M. Steyvers, and P. Smyth. The author-topic model for authors and documents. In *Proceedings of the Conference on Uncertainty in Artificial Intelligence*, volume 20, 2004.

[10] P. Smolensky. Information processing in dynamical systems: foundations of harmony theory. In D.E. Rumehart and J.L. McClelland, editors, *Parallel Distributed Processing: Explorations in the Microstructure of Cognition. Volume 1: Foundations*. McGraw-Hill, New York, 1986.

[11] M. Welling, F. Agakov, and C.K.I. Williams. Extreme components analysis. In *Advances in Neural Information Processing Systems*, volume 16, Vancouver, Canada, 2003.

[12] M. Welling and G.E. Hinton. A new learning algorithm for mean field Boltzmann machines. In *Proceedings of the International Conference on Artificial Neural Networks*, Madrid, Spain, 2001.

[13] M. Welling, G.E. Hinton, and S. Osindero. Learning sparse topographic representations with products of student-t distributions. In *Advances in Neural Information Processing Systems*, volume 15, Vancouver, Canada, 2002.

[14] R. Zemel, C. Williams, and M. Mozer. Lending direction to neural networks. *Neural Networks*, 8(4):503–512, 1995.
